# CLIP in Mirror: Disentangling text from visual images through reflection

**Tiancheng Wang**[1] **Yuguang Yang**[2] **Linlin Yang**[4*] **Shaohui Lin**[5] **Juan Zhang**[1,3]
**Guodong Guo**[6] **Baochang Zhang**[1,3]

[1]Institute of Artificial Intelligence, Beihang University, Beijing, China
[2]School of Electronic Information Engineering, Beihang University, Beijing, China
[3]Zhongguancun Laboratory, Beijing, China
[4]State Key Laboratory of Media Convergence and Communication,
Communication University of China, Beijing, China
[5]School of Computer Science and Technology, East China Normal University, Shanghai, China
[6]Ningbo Institute of Digital Twin, Eastern Institute of Technology, Ningbo, China

## Abstract

The CLIP network excels in various tasks, but struggles with text-visual images *i.e.*, images that contain both text and visual objects; it risks confusing textual and visual representations. To address this issue, we propose MirrorCLIP, a zero-shot framework, which disentangles the image features of CLIP by exploiting the difference in the mirror effect between visual objects and text in the images. Specifically, MirrorCLIP takes both original and flipped images as inputs, comparing their features dimension-wise in the latent space to generate disentangling masks. With disentangling masks, we further design filters to separate textual and visual factors more precisely, and then get disentangled representations. Qualitative experiments using stable diffusion models and class activation mapping (CAM) validate the effectiveness of our disentanglement. Moreover, our proposed MirrorCLIP reduces confusion when encountering text-visual images and achieves a substantial improvement on typographic defense, further demonstrating its superior ability of disentanglement. Our code is available at https://github.com/tcwangbuaa/MirrorCLIP.

## 1 Introduction

The CLIP network [19] has demonstrated remarkable success, leading to its widespread application in real-world scenarios. However, it still struggles with text-visual images [7; 16; 11; 1], *i.e.*, images that contain both text and visual objects, where CLIP can become confused when processing an image with misleading text in it. For instance, as shown in Figure 1, when asked to describe a visual object, the model might mistake an image of a dog for a cat due to the presence of the "cat" text. In contrast, when asked to recognize text in an image, the model might mistake the text of "eraser" for "egg" due to the visual object of eggs. Can we disentangle the textual and visual[1] factors within CLIP? Achieving this disentanglement for CLIP can reduce confusion when encountering such images and enhance its robustness against typographic attacks [1].

Existing work emphasizes extracting visual features from image features and exploring additional structures [16] or training strategies [1; 11]. Specifically, Materzynska et al. [16] introduce a learnable projection, Defense Prefix [1] introduces a learnable prefix in the prompt, and PAINT [11] introduces fine-tuning with linearly interpolating neural network weights. However, these methods require

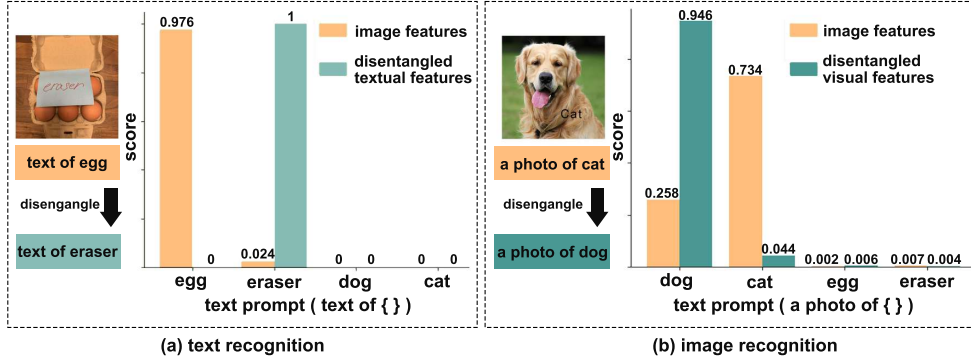

Figure 1: The zero-shot prediction of CLIP before and after disentanglement, (a): prediction of text recognition, text of "eraser" is misclassified as "egg" before disentanglement, (b): prediction of image recognition, visual object of a dog is misclassified as a cat before disentanglement
.

training with specified data and overlook textual features. Instead, we aim at a zero-shot architecture without retraining for CLIP and emphasize both textual and visual features via disentanglement.

In this paper, we propose MirrorCLIP, a simple yet efficient text-visual disentanglement framework to enhance CLIP's robustness in text-visual images. Our innovation is leveraging the differences of mirror effect between text and visual elements in the image - when images are horizontally flipped, visual objects maintain semantic consistency after flipping, while text typically becomes a nonsensical string. For instance, after being flipped, an image of a dog still remains its recognizability, while an image containing the word "cat" turns into a nonsensical string like "tac", resulting in the disappearance of its meaning. Based on this observation, we propose to decompose the image features of CLIP into textual and visual factors by contrasting them before and after flipping in the latent space. Specifically, MirrorCLIP employs a dual-stream zero-shot framework. The process begins by inputting both the original and horizontally flipped images into the image encoder to generate corresponding image features. By comparing these features, we generate a disentangling mask that identifies textual and visual regions of the latent variable. This mask is then used to separate textual and visual features. Specifically, the textual features are derived by excluding visual features from the original image features using the mask, while the visual features are obtained by combining image features of the original images with their flipped version.

Extensive experiments validate our proposed method. For text-visual disentanglement, the class activation maps (CAMs) [24] show that the disentangled textual and visual features correspond precisely to the regions of text and visual objects, respectively. Using the stable diffusion model [21; 20], visual features generate images similar to the original but without text, while textual features generate textual images (*i.e.*, images only contain text), demonstrating the effectiveness of our method. To quantitatively evaluate the effectiveness of visual feature disentanglement, we compared the state-of-the-art typographic defense methods Defense Prefix [1] in 10 synthetic and 3 real-world typographic attack datasets using disentangled features. Typographic attacks add text on top of visuals, testing the model's robustness against textual perturbations. MirrorCLIP achieves substantial performance improvements, with a +4.17% increase in real-world datasets and a +5.89% increase in synthetic datasets. To further evaluate the disentangled textual features, we propose to recognize the typographic attack text. The results show that with disentangled textual features, the accuracy improves to 73.95%, compared to 39.32% without disentanglement. In summary, the contributions of our work are as follows:

- We observed that CLIP exhibits horizontal flip invariance for the visual factors of images but not for the textual factors, and propose a simple yet efficient solution to disentangle textual features from visual features in the latent space of CLIP accordingly.

- We propose MirrorCLIP, a zero-shot text-visual disentanglement framework, which can effectively achieve the disentanglement of visual and textual features without any additional training and significantly reduce confusion in text-visual images while improving the robustness of CLIP against typographic attacks.

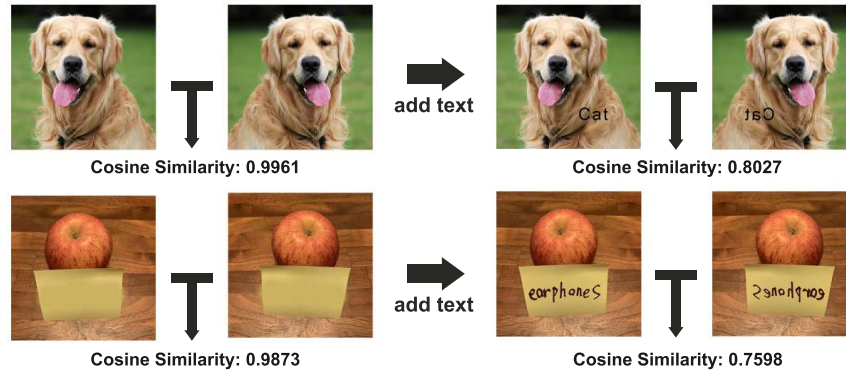

Figure 2: The cosine similarity of the image features encoded by the CLIP image encoder before and after horizontal flipping. Adding text to the image leads to a significant decrease in cosine similarity, indicating that CLIP does not exhibit horizontal flip invariance for textual factors.

- We qualitatively demonstrate the effectiveness of our disentangled representations through the salient regions of CAMs. Moreover, with stable diffusion models and our disentangled representations, we enable generation based on visual and textual factors.

- By evaluating on typographic images, we show that MirrorCLIP effectively achieves disentangled representations and greatly improves performance compared to CLIP without disentanglement, including a whopping 16.82% improvement on image recognition and 34.63% improvement on text recognition, surpassing state-of-the-art methods on defense against typographic attacks.

## 2   Related Work

**Vision-language models** have advanced significantly, learning generalized visual representations that align with textual descriptions [19]. This capability enables VLMs to make few-shot or zero-shot decisions in open-world settings [8; 25; 26], making them highly effective for downstream tasks. However, this broad generalization also raises concerns about robustness, especially when dealing with images containing rich text elements, which can mislead the model's decision results. MirrorCLIP further explores this setting, aiming to disentangle textual and visual features from text-visual images to improve CLIP's robustness in these challenging scenarios.

**Typographic attacks** were first introduced by Goh *et al.* [7], who revealed that the performance of vision-language models drops dramatically when input images contain misleading text. To mitigate this, Materzynska et al.[16] applied a linear projection matrix to disentangle visual from textual features. Ilharco et al.[11] interpolated between fine-tuned and original CLIP models, and Azuma et al. [1] introduced a learnable defense prefix. We utilize this task to evaluate the disentangled textual and visual features: visual features are used in typographic defense experiments, and textual features are used in typographic text recognition experiments.

**Disentangled representations of CLIP** have been studied to separate different types of information encoded in embeddings. Ramesh *et al.* [20] used PCA to reconstruct CLIP embeddings and generated related images through diffusion models, revealing distinct semantic dimensions. Lemesle *et al.* [14] found that textual and visual factors of an image do not share semantic representations in CLIP. Materzynska *et al.* [16] trained projection matrices to disentangle visual and textual features. Mirror-CLIP further explores the way to uncover the textual and visual components of the representations of text-visual images.

## 3   CLIP's Mirror Effect and Disentangling Masks

**Contrastive Language-Image Pretraining.** CLIP [19] aims to learn robust associations between text and images without requiring explicit labeling or supervision for specific tasks. It is pretrained on a dataset including 400 million image-text pairs without human annotation, which provides a broad spectrum of possible text-image associations. During training, through contrastive learning,

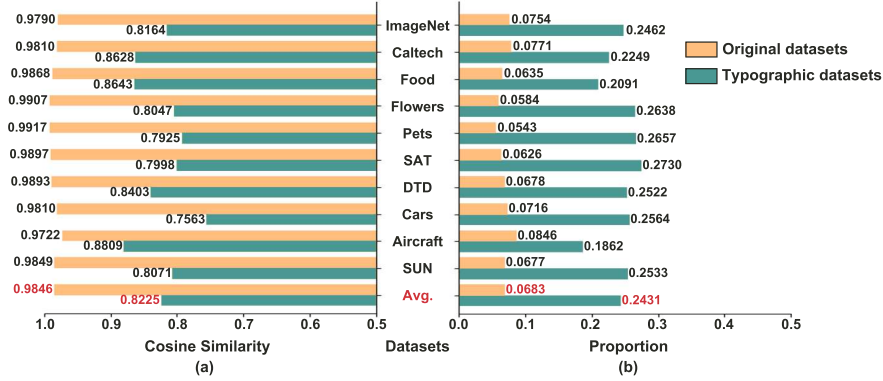

Figure 3: Results of mirror effect experiments, (a) Cosine similarity of image features before and after flipping on Original and Typographic datasets, (b) Proportion of textual mask on Original and Typographic datasets.

CLIP optimizes to maximize the cosine similarity of embeddings between matching text-image pairs. This enables CLIP to learn the embeddings of images and text within a joint latent space, thereby allowing CLIP to extract the semantics of images. However, recent work has revealed that CLIP can become confused when faced with text-visual images [7; 16; 11; 1]. To address this issue, we leverage CLIP's mirror effect to achieve the disentanglement of textual and visual components within the image embeddings.

**CLIP's Mirror Effect.** When we observe objects in the mirror, we are able to identify their reflected presence. However, this may not be the case with text. Because the text that is mirrored appears as a string of non-sensical characters due to the letter distortion and the reversed writing order. CLIP is a joint image and text embedding model designed to recognize concepts in images. Does CLIP act like a human and exhibit a similar phenomenon?

To determine whether the distinct mirror effects between visual objects and text affect the representation of CLIP, we input both the original and flipped images into the encoder and calculate the cosine similarity between the resulting features. As shown in Figure 2, for clean input images, such as a dog, the cosine similarity remains approximately 1, indicating semantic invariance. However, when text is added, the similarity between the original and flipped images significantly decreases. Furthermore, our quantitative experiments on 10 public datasets reveal that similarity drops significantly from 0.9846 to 0.8225 once text is added to the images as shown in Figure 3 (a). This demonstrates that the image features of visual objects in CLIP have horizontal flip invariance, whereas that of text does not, which can be further exploited to disentangle these two factors from image representations.

**Disentangling Mask.** Given $X$ and $X^f$ represent the image features before and after image flipping, their cosine similarity can be written as:

$$cos\left(X, X^f\right) = \frac{\sum_{i=1}^n \left(X_i \odot X_i^f\right)}{\|X\| \times \|X^f\|} = \sum_{i=1}^n \left(\frac{X_i}{\|X\|} \odot \frac{X_i^f}{\|X^f\|}\right), \tag{1}$$

where $\odot$ denotes Hadamard product, $n$ denotes the dimensionality of features. The cosine similarity is influenced by the product of elements at different positions after normalization. If the signs of elements at the corresponding positions change after flipping, the product becomes negative, leading to a decrease in cosine similarity. Previous research [20] has shown that different dimensions of CLIP's image embeddings encode distinct semantic information. Therefore, we use the change in the signs of elements at different positions before and after flipping to determine whether a position belongs to the textual or visual factors. Subsequently, we generate the disentangling mask for textual and visual factors with this characteristic. As shown in Figure 4 (a), the disentangling mask $M$, and its corresponding textual mask $M^t$ and visual mask $M^v$, are calculated as

$$M = Sign\left(X \odot X^f\right), \tag{2}$$

$$M^t = -\frac{1}{2} \times (M - 1), \quad M^v = \frac{1}{2} \times (M + 1), \tag{3}$$

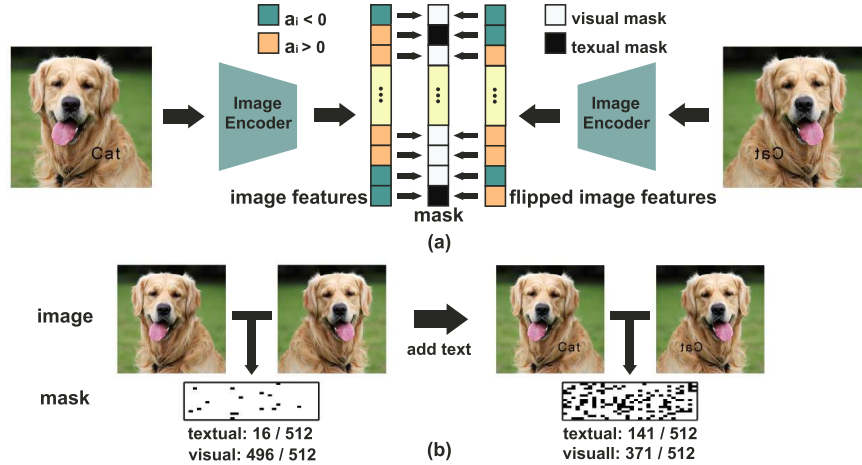

Figure 4: Generation of disentangling mask, (a) The disentangling mask is generated by contrasting the sign of corresponding positions in the image features before and after flipping. (b) Input images and generated disentangling masks (resized from $1 \times 512$ to $16 \times 32$). After adding text to the image, the proportion of textual mask increases significantly.

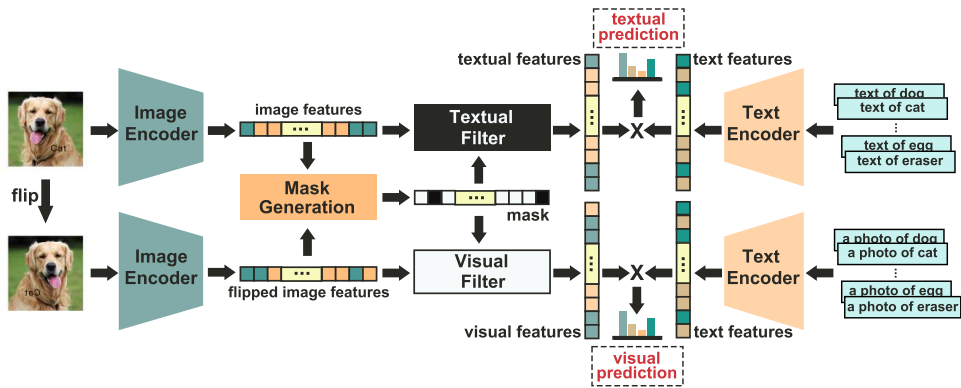

Figure 5: Pipeline of zero-shot dual-stream disentanglement framework. The framework takes flipped and original images as input, generates disentangling masks by comparing their image features in the latent space, then utilizes the proposed textual filter and visual filter to generate textual and visual features, achieving disentanglement and completing downstream tasks.

$M^t$ and $M^v$ are obtained by mapping $\{-1, 1\}$ in $M$ to $\{1, 0\}$ and $\{0, 1\}$, respectively. Figure 4 (b) shows the disentangling masks for different images, where the black areas represent the textual mask and the white areas represent the visual mask. It can be observed that when text is added to the image, whether handwritten or printed, the area of the textual mask increases significantly. As shown in Figure 3 (b), we conduct experiments across 10 public datasets and reveal that the proportion of textual mask increases significantly from 0.0683 to 0.2431 after adding text to images, which confirms the validity of our mask generation method.

## 4 Zero-shot Disentanglement Framework

Utilizing the disentangling masks, we propose a zero-shot dual-stream disentanglement framework. The pipeline of the disentanglement framework is illustrated in Figure 5. The framework is straightforward and does not require any training. We generate the disentangling mask by comparing the image features before and after image flipping based on Eqs. 2 and 3. Due to the entangling between textual and visual features, any dimension of latent space may contain both textual and visual factors. Therefore, separating with a "boolean" disentangling mask is rough. For example, directly setting the

visual mask area to 0 would lead to the loss of the textual semantic information within it. Instead, we propose a "soft" textual filter to remove visual features in the image features as below:

$$X^t = (X \odot M^t) + (X - X^f) \odot M^v, \qquad (4)$$

where $X^t$ denotes textual features. The textual filter in Equation 4 consists of two parts: the first part ensures the retention of textual features in the textual mask region, and the second part preserves textual features in the visual mask region while filtering out visual features.

Similarly, we propose a visual filter to get visual features $X^v$ as below:

$$X^v = X^f + X \odot M^v. \qquad (5)$$

In Equation 5, the first part uses flipped image features as visual features due to the disappearance of textual semantics after flipping, and the second part enhances robustness against images with flipped text by adding a visual mask region of original image features. After disentangling, either textual features or visual features can be used to replace original image features for inference depending on the specific task. As shown in Figure 1, our disentanglement framework can correct errors made by CLIP in image and text recognition.

## 5 Experiment

### 5.1 Experimental Setup

**Overview.** To validate the effectiveness of the proposed MirrorCLIP, we conduct the following experiments. In Section 5.2, we first validate the difference in the mirror effect of text and visual elements in images with 10 clean public datasets and their corresponding synthetic typographic datasets. We then qualitatively visualize the quality of disentangled text and visual elements using CAMs and stable diffusion models. In Section 5.3, we primarily validate the disentanglement effectiveness for visual elements by performing a typographic defense experiment on 13 typographic datasets (10 synthetic datasets and 3 real-world datasets), following [1]. In Section 5.4, we evaluate the disentanglement effectiveness for text elements by ensuring that the disentangled textual features can correctly recognize the text added to visual elements in the typographic datasets.

**Datasets. Clean public classification datasets** contain rich visual elements from the real world, which can be used to evaluate the robustness and performance of MirrorCLIP. These include ImageNet [4], Caltech101 [6], OxfordPets [18], StanfordCars [13], Flowers102 [17], Food101 [2], FGVCAircraft [15], DTD [3], SUN397 [23], and EuroSAT [10]. **Synthetic typographic Datasets** add text of incorrect categories to the images. We follow [1] to construct synthetic typographic datasets using the 10 clean public datasets mentioned above. **Real-world typographic datasets** include three publicly available real-world typographic attack datasets from Materzynska et al. [16], PAINT[11], and Defense Prefix (RTA-100) [1].

**Baselines.** To evaluate MirrorCLIP 's disentanglement performance for visual elements, we benchmark against CLIP [19], Materzynska *et al.* [16], PAINT [11] and Defense Prefix [1]. To evaluate MirrorCLIP 's disentanglement performance for text elements, we mainly compare it with the vanilla CLIP.

### 5.2 Validation Experiments for MirrorCLIP

**Validation of different mirror effects for visual and text elements.** To validate our observation that CLIP exhibits horizontal flip invariance for visual features but not for textual features, we conducted experiments on the cosine similarity of image features before and after flipping across 10 clean public image classification datasets and their corresponding typographic datasets.

The average cosine similarity of image features before and after flipping for all samples in all datasets is shown in Table 1. According to the results, before adding text, the cosine similarity of image features before and after flipping is $0.9846$ across 10 datasets, which is close to 1 and confirms CLIP's horizontal flip invariance for visual features. However, after adding text, the cosine similarity significantly decreases to $0.8225$, which also validates CLIP's lack of horizontal flip invariance for textual features.

Table 1: Cosine similarity of image features before and after flipping on Clean and Typographic datasets.

|  | ImageNet | Caltech | Food | Flowers | Pets | SAT | DTD | Cars | Aircraft | SUN | Avg. |
|---|---|---|---|---|---|---|---|---|---|---|---|
| Original | 0.9790 | 0.9810 | 0.9868 | 0.9907 | 0.9917 | 0.9897 | 0.9893 | 0.9810 | 0.9722 | 0.9849 | 0.9846 |
| Typographic | 0.8164 | 0.8628 | 0.8643 | 0.8047 | 0.7925 | 0.7998 | 0.8403 | 0.7563 | 0.8809 | 0.8071 | 0.8225 |

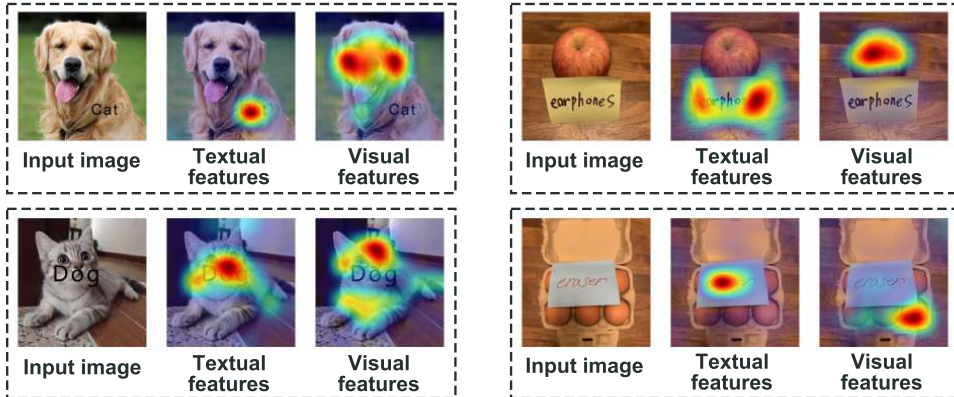

Figure 6: Visualization for textual and non-textual features for typographic attacked data using class activation map.

**Visualization of attribution maps for textual and visual features.** We employ CAM to showcase a more intuitive visualization towards the disentangled textual and visual features in Equations 4 and 5, respectively. CAMs illuminate the salient regions within an input image that contribute the most strongly to the target concept [22]. Specifically, we employ DecomCAM [24], which is the state-of-the-art CAM method that strongly reduces the noise effect of typical CAM methods. Our interpretation target is constructed as $X^t X^T$, $X^c X^T$ for textual and visual features, respectively. As shown in Figure 6, our disentangled features can effectively separate regions of text and visual elements.

**Similar images generation with textual and visual features.** To visualize the effectiveness of disentanglement, we utilized Stable Diffusion models, Stable UnCLIP [20], with disentangled textual and visual features for image generation [20; 16]. We use image features, disentangled visual and textual features as image embedding conditions. The generated images are depicted in Figure 7.

When using images with irrelevant text for image variations, the generated images mix the semantics of text and visual elements. For example, an image of an apple with the text "earphones" might produce an image with the logo of Apple Inc. and non-sensical text. Similarly, a cat labeled as "dog" could result in an image that combines a cat and a dog's face with non-sensical text. Besides, after disentangling, visual features generate accurate visual content (*e.g.*, an apple or a cat) without non-sensical strings, indicating successful filtering of textual features. Textual features, on the other hand, generate images filled with text without visual semantics, demonstrating effective separation.

In addition, we conducted controlled experiments. for images without text, visual features generate image-related patterns, while textual features produce nonsensical characters. For text-only images, visual features generate non-sensical patterns, whereas textual features generate coherent text and patterns, showcasing the effectiveness of our disentanglement framework.

## 5.3 Evaluation on visual features disentanglement

To further evaluate the effectiveness of the disentangled visual features, we perform typographic defense experiments. Here, MirrorCLIP is required to exclude the interference of the text added and correctly recognize visual elements. The performance of visual features on clean public classification

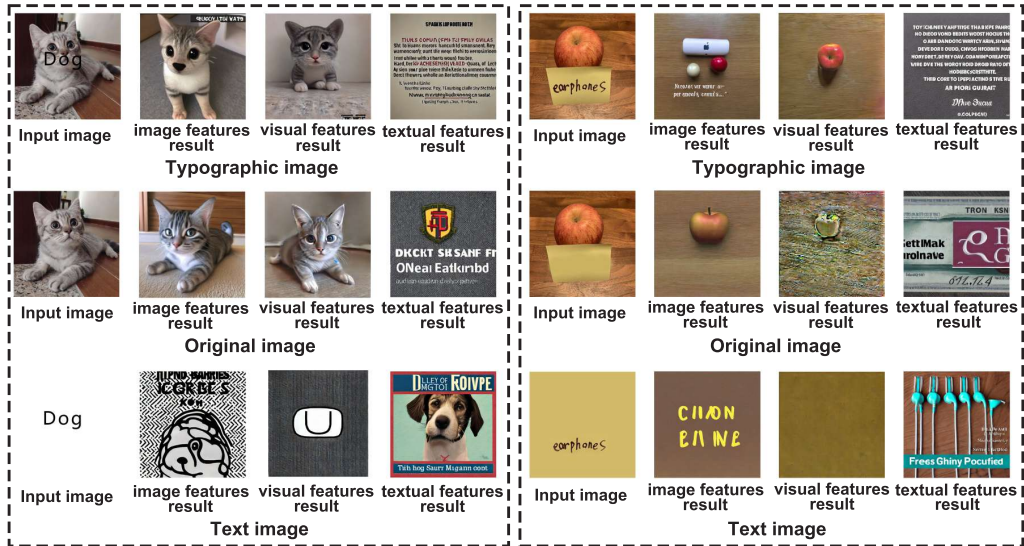

Figure 7: Results of image variation using Stable unCLIP.

Table 2: Results of image classification on original datasets.

|  | ImageNet | Caltech | Food | Flowers | Pets | SAT | DTD | Cars | Aircraft | SUN | Avg. |
|---|---|---|---|---|---|---|---|---|---|---|---|
| CLIP | 62.05 | 88.69 | 84.13 | 66.32 | 87.38 | 43.10 | 44.68 | 58.71 | 19.11 | 61.70 | 61.59 |
| Materzynska+ [16] | 54.38 | 80.53 | 55.01 | 51.86 | 75.01 | 37.32 | 36.28 | 40.33 | 13.23 | 51.06 | 49.50 |
| PAINT [11] | 61.82 | 88.48 | 80.51 | 64.73 | 85.23 | 38.20 | 42.61 | 55.30 | 17.73 | 61.69 | 59.63 |
| Defense Prefix [1] | **62.48** | **89.28** | 83.65 | 63.82 | 87.22 | **43.85** | 40.64 | 57.47 | 19.26 | 61.41 | 60.91 |
| Ours | 62.34 | 89.17 | **84.52** | **66.34** | **87.71** | 43.77 | **45.00** | **59.07** | **19.41** | **62.12** | **61.95** |

datasets, synthetic typographic datasets, and real-world typographic datasets is shown in Tables 2, 3, and 4, respectively.

From Tables 3 and 4, it is evident that CLIP exhibits poor robustness when faced with typographic attacks, resulting in significant performance degradation. However, using the visual features obtained from our disentanglement framework to replace the original image features significantly improves performance on both synthetic and real-world typographic attack datasets ($+15.49\%$ on synthetic typographic attack datasets and $+21.27\%$ on real-world typographic attack datasets). Compared to Materzynska *et al.* [16], PAINT [11], and Defense Prefix [1], which introduce additional parameters for training, our zero-shot method surpasses their performance without any additional training. This demonstrates the strong robustness of the visual features obtained from our simple but effective disentanglement framework across various types of images.

Additionally, we test the performance of disentangled visual features on clean datasets. According to Table 2, our zero-shot disentanglement framework slightly improves performance compared to the original CLIP model in clean images without text. This indicates that our method does not cause performance degradation when handling images without text elements.

## 5.4 Evaluation on textual features disentanglement

To evaluate the performance of the textual features obtained from our framework, we employ these features to recognize the text elements in the typographic datasets. Our results are shown in Table 5. Based on the results, CLIP struggles to achieve high performance in text recognition within images due to the confusion between text and visual elements. However, after applying our disentanglement framework, substituting image features with textual features significantly improves CLIP's performance in text recognition (from $39.32\%$ to $73.95\%$). This indicates that the disentangled textual features can precisely represent the text elements in the images, confirming the effectiveness of the proposed framework.

Table 3: Results of image classification on synthetic typographic attack datasets.

|                    | ImageNet | Caltech | Food  | Flowers | Pets  | SAT   | DTD   | Cars  | Aircraft | SUN   | Avg.  |
|--------------------|----------|---------|-------|---------|-------|-------|-------|-------|----------|-------|-------|
| CLIP               | 39.28    | 64.16   | 57.20 | 31.06   | 59.12 | 4.77  | 24.73 | 20.57 | 10.68    | 34.43 | 34.60 |
| Materzynska+ [16]  | 44.91    | 74.73   | 43.41 | 34.95   | 63.61 | 16.22 | 33.03 | 15.79 | 8.28     | 39.52 | 37.44 |
| PAINT [11]         | **55.90** | 83.57   | 72.94 | **54.92** | 76.53 | 17.31 | **36.60** | **33.44** | 14.46  | 53.62 | 49.93 |
| Defense Prefix [1] | 49.83    | 79.54   | 67.79 | 44.12   | 72.88 | 9.65  | 31.60 | 28.64 | **14.49** | 43.50 | 44.20 |
| Ours               | 52.72    | **84.18** | **76.30** | 50.11 | **76.70** | **25.44** | 35.32 | 32.53 | 13.86 | **53.74** | **50.09** |

Table 4: Results of image classification on real-world typographic attack datasets.

|                    | from [16] | from [11] | RTA-100 [1] | Avg.  |
|--------------------|-----------|-----------|-------------|-------|
| CLIP               | 45.56     | 50.00     | 46.70       | 47.42 |
| Materzynska+ [16]  | **77.78** | 55.45     | 57.60       | 63.61 |
| PAINT [11]         | 53.22     | 58.18     | 53.60       | 55.00 |
| Defense Prefix [1] | 71.93     | 63.64     | 58.00       | 64.52 |
| Ours               | 67.27     | **73.89** | **64.90**   | **68.69** |

Table 5: Results of text recognition on typographic attack datasets.

|      | synthetic | | | real-world | | | Avg. |
|------|-----------|---------|-------|-----------|-----------|---------|------|
|      | Imagenet  | Flowers | Food  | from [16] | from [11] | RTA-100 |      |
| CLIP | 30.67     | 61.60   | 34.48 | 26.11     | 42.73     | 40.30   | 39.32 |
| Ours | **64.06** | **81.22** | **80.91** | **72.78** | **75.45** | **69.30** | **73.95** |

Table 6: Results of different features on various tasks.

|                        | image recognition | | text recognition |
|------------------------|----------|------------|------------------|
|                        | original | typographic | real-world       |
| image features         | 61.59    | 37.56      | 36.38            |
| flipped image features | 61.38    | **55.97**  | 0.57             |
| textual features       | 3.53     | 0.78       | **72.51**        |
| textual features (hard)| 2.10     | 0.77       | 61.03            |
| visual features        | **61.95** | 54.38     | 5.29             |
| visual features (hard) | 61.69    | 45.97      | 23.18            |

## 5.5 Analysis and Ablation

**Structure of image encoder.** To investigate whether the horizontal flip invariance is a characteristic of ViT [5], we replaced CLIP's image encoder with ResNet [9] (RN50×4) and conducted the same experiment; the experimental results confirm that our observations and disentanglement framework are also applicable to CLIP models using CNN-based image encoders. This indicates that the characteristic of horizontal flip invariance is mainly related to the contrastive learning strategy employed by CLIP.

**Filter of textual and visual features.** In order to figure out the appropriate representation of textual and visual features, we conducted experiments on various tasks using different feature representations obtained after disentanglement. The results of the experiments are shown in Table 6, where "original" represents the average accuracy on 10 original image classification datasets, "typographic" represents the average accuracy on 13 real-world and synthetic typographic datasets, "real-world" represents the average accuracy on 3 real-world typographic datasets. Textual features (hard) and visual features (hard) are obtained through $X \odot M^t$ and $X \odot M^v$, respectively, where $X$ denotes image features, $M^t$ and $M^v$ represent the textual mask and the visual mask.

According to Table 6, as analyzed in Section 4, directly zeroing out the textual or visual parts based on the disentangling mask would lead to information loss and consequently performance degradation. So we proposed the textual filter in Equation 4 to obtain textual features, aiming to reduce information loss. Although directly using flipped image features as visual features achieve the highest performance in defending against typographic attacks, it lacks robustness for flipped images (*i.e.*, if all samples are flipped before being inputted, the accuracy would decrease to $37.56\%$, detailed results are shown in

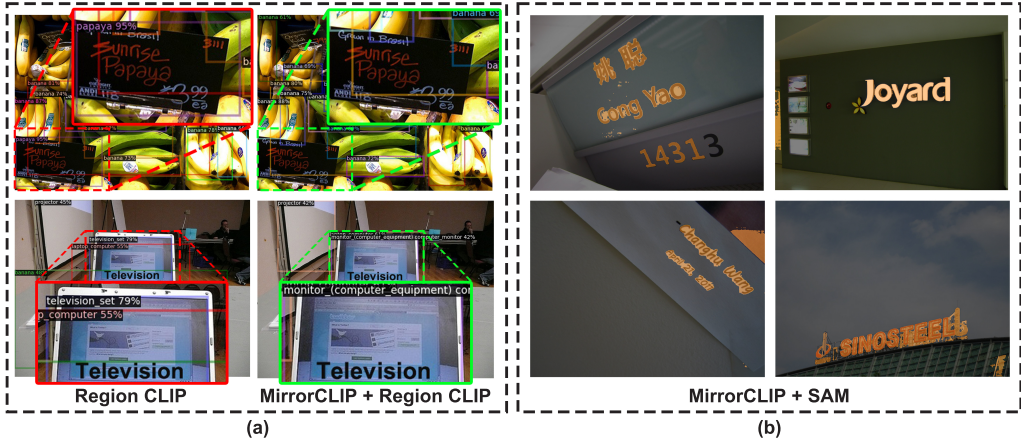

Figure 8: Potential Application Examples of MirrorCLIP. (a) Using MirrorCLIP to disentangle region features of RegionCLIP, before disentanglement, RegionCLIP mistakenly identified a price tag with text "papaya" as papaya and a laptop monitor as a television set because of the interference of text "television". (b) Textual features disentangled by MirrorCLIP are used to provide prompts for SAM, achieving text region segmentation.

Table L). Therefore, we proposed the visual filter in Equation 5 to obtain visual features, aiming to ensure robustness against flipped images.

Moreover, as seen in Table 6, while textual features notably boost text recognition, they yield negligible accuracy in image recognition tasks. Conversely, visual features significantly enhance image recognition but have minimal impact on text recognition, which validates the effectiveness of our proposed filter in isolating visual and textual features.

## 6 Limitations and Conclusion

**Limitations.** Although our proposed framework achieves excellent disentanglement results with a simple approach, due to the deep entanglement between visual and textual features, our method cannot fully separate them. It does not affect performance in recognition tasks but may influence the results of image generation, as seen in Figure 7 with the examples of apple in the second row and textual features results in the first row. What's more, when facing extreme scenarios such as palindromes in the images, MirrorCLIP still work for normal palindromes, where the shape of the words changes before and after flipping (e.g., "did" to "bib"). However, for special palindromes, where the shape of the words remains basically unchanged (e.g., "mom" to "mom"), MirrorCLIP struggles to achieve disentanglement, although special palindromes are quite rare compared to other word, detailed experimental results are shown in Appendix F.

**Potential application.** We have initially explored object detection and text segmentation by combining MirrorCLIP with RegionCLIP [27] and SAM [12]. The results show the potential of MirrorCLIP for different downstream tasks or applications. Relevant examples are shown in Figure 8. By using MirrorCLIP to get the disentangled visual region features of RegionCLIP, we can reduce the influence of textual factors and get more accurate detection results. By using the textual features obtained from MirrorCLIP to generate prompts for SAM, we can achieve text localization within images and perform preliminary text segmentation.

**Conclusion.** In this paper, we first discovered and verified that CLIP exhibits horizontal flip invariance for visual features while lacking this property for textual features. Leveraging this observation, we proposed a simple yet effective zero-shot dual-stream disentanglement framework MirrorCLIP by contrasting image features before and after flipping. We demonstrated the effectiveness of this framework through the visualization of attention maps with CAMs and similar image generation with stable diffusion models. Additionally, we conducted experiments on 13 synthetic and real-world typographic attack datasets to further validate the excellent disentanglement performance and robustness of our method across different samples. Furthermore, we surpass state-of-the-art methods in defense against typographic attacks without any additional training.

## Acknowledgements

The work is supported by the National Key Research and Development Program of China (Grant No. 2023YFC3306401). This research was also supported by National Natural Science Foundation of China (No. 12201024), Zhejiang Provincial Natural Science Foundation of China under Grant No. LD24F020007, Beijing Natural Science Foundation (No. L223024 and L244043), "One Thousand Plan" projects in Jiangxi Province (JXSQ2023102268).

## Footnotes

*Corresponding Author, `lyang@cuc.edu.cn`

[1]For simplify, we refer to 'visual' as the opposite of 'textual', *i.e.*, 'non-textual', for text-visual images.

## References

[1] Hiroki Azuma and Yusuke Matsui. Defense-prefix for preventing typographic attacks on clip. In *IEEE/CVF International Conference on Computer Vision Workshops (ICCVW)*, 2023.

[2] Lukas Bossard, Matthieu Guillaumin, and Luc Van Gool. Food-101 - mining discriminative components with random forests. In *European Conference on Computer Vision (ECCV)*, 2014.

[3] Mircea Cimpoi, Subhransu Maji, Iasonas Kokkinos, Sammy Mohamed, and Andrea Vedaldi. Describing textures in the wild. In *IEEE/CVF Conference on Computer Vision and Pattern Recognition (CVPR)*, 2013.

[4] Jia Deng, Wei Dong, Richard Socher, Li-Jia Li, K. Li, and Li Fei-Fei. Imagenet: A large-scale hierarchical image database. In *IEEE/CVF Conference on Computer Vision and Pattern Recognition (CVPR)*, 2009.

[5] Alexey Dosovitskiy, Lucas Beyer, Alexander Kolesnikov, Dirk Weissenborn, Xiaohua Zhai, Thomas Unterthiner, Mostafa Dehghani, Matthias Minderer, Georg Heigold, Sylvain Gelly, Jakob Uszkoreit, and Neil Houlsby. An image is worth 16x16 words: Transformers for image recognition at scale. In *International Conference on Learning Representations (ICLR)*, 2021.

[6] Li Fei-Fei, Rob Fergus, and Pietro Perona. Learning generative visual models from few training examples: An incremental bayesian approach tested on 101 object categories. In *IEEE/CVF Conference on Computer Vision and Pattern Recognition Workshop (CVPRW)*, 2004.

[7] Gabriel Goh, Nick Cammarata, Chelsea Voss, Shan Carter, Michael Petrov, Ludwig Schubert, Alec Radford, and Christopher Olah. Multimodal neurons in artificial neural networks. *Distill*, 2021.

[8] Jie Guo, Qimeng Wang, Yan Gao, Xiaolong Jiang, Xu Tang, Yao Hu, and Baochang Zhang. Mvp-seg: Multi-view prompt learning for open-vocabulary semantic segmentation. In *Chinese Conference on Pattern Recognition and Computer Vision (PRCV)*, 2023.

[9] Kaiming He, X. Zhang, Shaoqing Ren, and Jian Sun. Deep residual learning for image recognition. In *IEEE/CVF Conference on Computer Vision and Pattern Recognition (CVPR)*, 2015.

[10] Patrick Helber, Benjamin Bischke, Andreas R. Dengel, and Damian Borth. Eurosat: A novel dataset and deep learning benchmark for land use and land cover classification. *IEEE Journal of Selected Topics in Applied Earth Observations and Remote Sensing*, 2017.

[11] Gabriel Ilharco, Mitchell Wortsman, Samir Yitzhak Gadre, Shuran Song, Hannaneh Hajishirzi, Simon Kornblith, Ali Farhadi, and Ludwig Schmidt. Patching open-vocabulary models by interpolating weights. In *Advances in Neural Information Processing Systems (NeurIPS)*, 2022.

[12] Alexander Kirillov, Eric Mintun, Nikhila Ravi, Hanzi Mao, Chloe Rolland, Laura Gustafson, Tete Xiao, Spencer Whitehead, Alexander C. Berg, Wan-Yen Lo, Piotr Dollár, and Ross B. Girshick. Segment anything. In *IEEE/CVF International Conference on Computer Vision (ICCV)*, pages 3992–4003, 2023.

[13] Jonathan Krause, Michael Stark, Jia Deng, and Li Fei-Fei. 3d object representations for fine-grained categorization. In *IEEE International Conference on Computer Vision Workshops (ICCVW)*, 2013.

[14] Yoann Lemesle, Masataka Sawayama, Guillermo Valle Pérez, Maxime Adolphe, Hélène Sauzéon, and Pierre-Yves Oudeyer. Language-biased image classification: evaluation based on semantic representations. In *International Conference on Learning Representations (ICLR)*, 2022.

[15] Subhransu Maji, Esa Rahtu, Juho Kannala, Matthew B. Blaschko, and Andrea Vedaldi. Fine-grained visual classification of aircraft. *ArXiv*, 2013.

[16] Joanna Materzynska, Antonio Torralba, and David Bau. Disentangling visual and written concepts in clip. In *IEEE/CVF Conference on Computer Vision and Pattern Recognition (CVPR)*, 2022.

[17] Maria-Elena Nilsback and Andrew Zisserman. Automated flower classification over a large number of classes. In *Indian Conference on Computer Vision, Graphics & Image Processing*, 2008.

[18] Omkar M. Parkhi, Andrea Vedaldi, Andrew Zisserman, and C. V. Jawahar. Cats and dogs. In *IEEE/CVF Conference on Computer Vision and Pattern Recognition (CVPR)*, 2012.

[19] Alec Radford, Jong Wook Kim, Chris Hallacy, Aditya Ramesh, Gabriel Goh, Sandhini Agarwal, Girish Sastry, Amanda Askell, Pamela Mishkin, Jack Clark, Gretchen Krueger, and Ilya Sutskever. Learning transferable visual models from natural language supervision. In *International Conference on Machine Learning (ICML)*, 2021.

[20] Aditya Ramesh, Prafulla Dhariwal, Alex Nichol, Casey Chu, and Mark Chen. Hierarchical text-conditional image generation with clip latents. *ArXiv*, 2022.

[21] Robin Rombach, A. Blattmann, Dominik Lorenz, Patrick Esser, and Björn Ommer. High-resolution image synthesis with latent diffusion models. In *IEEE/CVF Conference on Computer Vision and Pattern Recognition (CVPR)*, 2021.

[22] Ramprasaath R. Selvaraju, Abhishek Das, Ramakrishna Vedantam, Michael Cogswell, Devi Parikh, and Dhruv Batra. Grad-cam: Visual explanations from deep networks via gradient-based localization. *International Journal of Computer Vision (IJCV)*, 2016.

[23] Jianxiong Xiao, Krista A. Ehinger, James Hays, Antonio Torralba, and Aude Oliva. Sun database: Exploring a large collection of scene categories. *International Journal of Computer Vision (IJCV)*, 2014.

[24] Yuguang Yang, Runtang Guo, Sheng Wu, Yimi Wang, Linlin Yang, Bo Fan, Jilong Zhong, Juan Zhang, and Baochang Zhang. Decomcam: Advancing beyond saliency maps through decomposition and integration. *Neurocomputing*, 2024.

[25] Yuguang Yang, Yiming Wang, Shupeng Geng, Runqi Wang, Yiming Wang, Shen-Te Wu, and Baochang Zhang. Self-enhancement improves text-image retrieval in foundation visual-language models. *ArXiv*, 2023.

[26] Renrui Zhang, Zhang Wei, Rongyao Fang, Peng Gao, Kunchang Li, Jifeng Dai, Yu Jiao Qiao, and Hongsheng Li. Tip-adapter: Training-free adaption of clip for few-shot classification. In *European Conference on Computer Vision (ECCV)*, 2022.

[27] Yiwu Zhong, Jianwei Yang, Pengchuan Zhang, Chunyuan Li, Noel C. F. Codella, Liunian Harold Li, Luowei Zhou, Xiyang Dai, Lu Yuan, Yin Li, and Jianfeng Gao. Regionclip: Region-based language-image pretraining. In *IEEE/CVF Conference on Computer Vision and Pattern Recognition (CVPR)*, pages 16772–16782, 2022.

## A  Implementation details

**Benchmark model.** During the experiments, we used the ViT-B/32 version of CLIP as a pre-trained model and all parameters of CLIP were frozen. We informed the CLIP model of our recognition intent by adjusting the text prompt. For text recognition, we used the template "text of {}" across all datasets. For image recognition, we used the template "a photo of {}" across all real-world typographic attack datasets, which is shown in Figure 5, and the templates we use across synthetic typographic attack datasets are shown in Table A. All experiments were conducted on NVIDIA A800.

Table A: Templates of synthetic typographic attack datasets for image recognition

| Dataset | template |
|---|---|
| ImageNet | "a photo of a {}" |
| Caltech101 | "a photo of a {}" |
| Food101 | "a photo of a {}, a type of food" |
| Flowers102 | "a photo of a {}, a type of flower" |
| OxfordPets | "a photo of a {}, a type of pet" |
| EuroSAT | "a centered satellite photo of a {}" |
| DTD | "{} texture" |
| StanfordCars | "a photo of a {}" |
| FGVCAircraft | "a photo of a {}, a type of aircraft" |
| SUN397 | "a photo of a {}" |

**Generating similar images for validation.** The model we employed for image generation in Section 5.2 is Stable unCLIP [20], a new stable diffusion model fine-tuned at $768 \times 768$ resolution, based on SD2.1-768 [21]. This model allows for image variations, conditioned on CLIP image features. During the image variation, we use '' as prompt condition.

## B  Typographic attack datasets

We will explain the details of the test data in Section 5. For synthetic typographic attack datasets, we add text to images from ten classification datasets: ImageNet [4], Caltech101 [6], OxfordPets [18], StanfordCars [13], Flowers102 [17], Food101 [2], FGVCAircraft [15], DTD [3], SUN397 [23], EuroSAT [10]. To generate typographic attack datasets, we followed PAINT [11] and Defense Prefix [1], as shown in Figure A (a). We resize the short dimension to 224 pixels using bicubic interpolation and center-crop the images by 224×224. We randomly select the font from three options: Roman, Courier, and Times. The font size is chosen randomly between 20 and 40 points. Additionally, we use one of eight colors: red, green, blue, cyan, magenta, yellow, white, or black. The text includes a 1-point shadow in a different color from the main font color. Text is placed randomly in the image, ensuring that entire words remain visible. The text content is chosen from the class labels of the dataset, excluding the correct labels for the images. The samples of synthetic typographic attack datasets are shown in Figure B.

For real-world typographic attack datasets, we use datasets made by Materzynska et al. [16], PAINT [11] and Defense Prefix [1]. The samples of real-world typographic attack datasets are shown in Figure C, in which objects are labeled with tags of incorrect classes.

## C  Visualization of attention map

More visualization results using CAMs to show attention maps in Section 5.2 are presented in Figure D.

## D  Generation of similar images

More similar images generated in Section 5.2 are shown in Figure E.

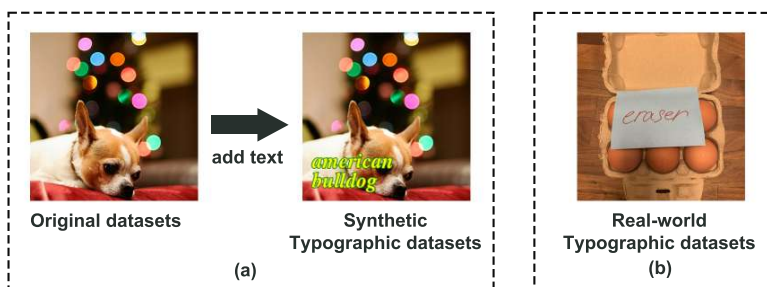

Figure A: Typographic datasets. (a) generation of synthetic typographic datasets. (b) a sample of real typographic datasets.

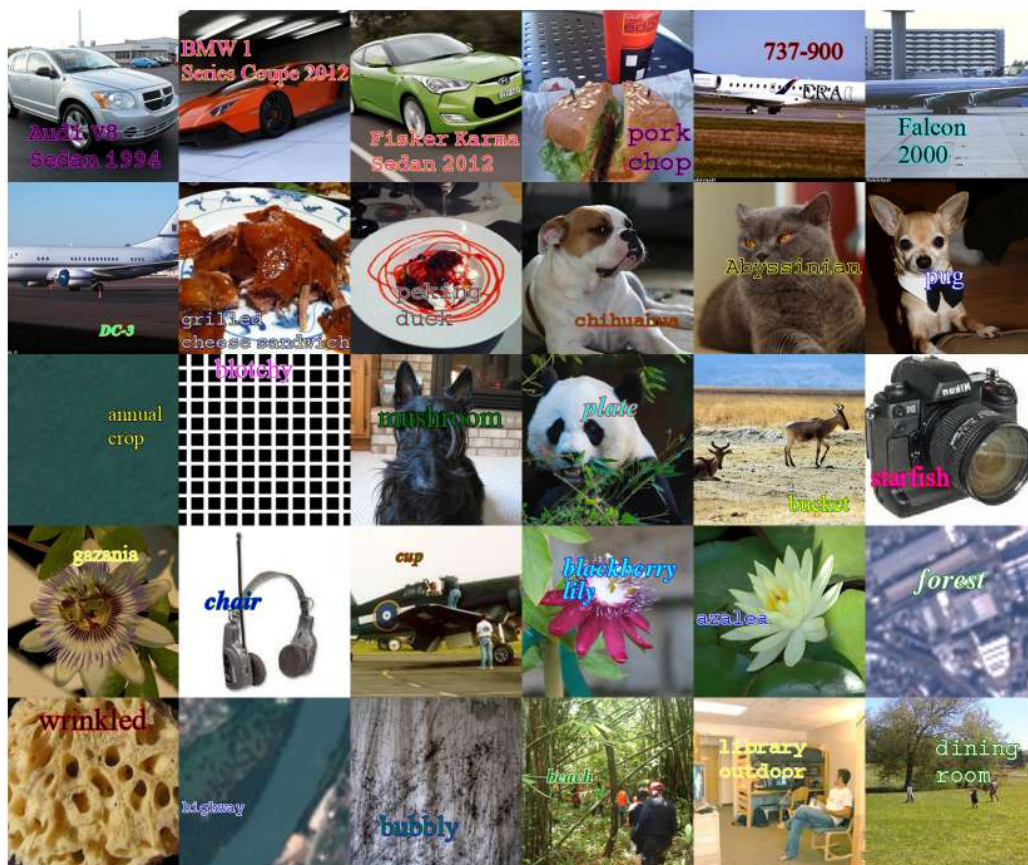

Figure B: Samples of synthetic typographic attack datasets.

# E    Detailed results of ablation

**Results of CNN-based image encoder.** The detailed results of the experiments for the CNN-based image encoder in Section 5.5 are as follows. The cosine similarity of the image features before and after the flip of the Clean and Typographic datasets is shown in Table B. The results of image recognition across 10 original datasets are shown in Table C. Results of image recognition across 10 synthetic typographic attack datasets and 3 real-world typographic attack datasets are shown in Table D and Table E, respectively. Results of text recognition across 3 synthetic typographic attack datasets and 3 real-world typographic attack datasets are shown in Table F.

**Results of different feature representations.** The detailed results of experiments we conduct for different feature representations in Section 5.5 are as follows. Results of image recognition across 10 original datasets are shown in Table G. Results of image recognition across 10 synthetic

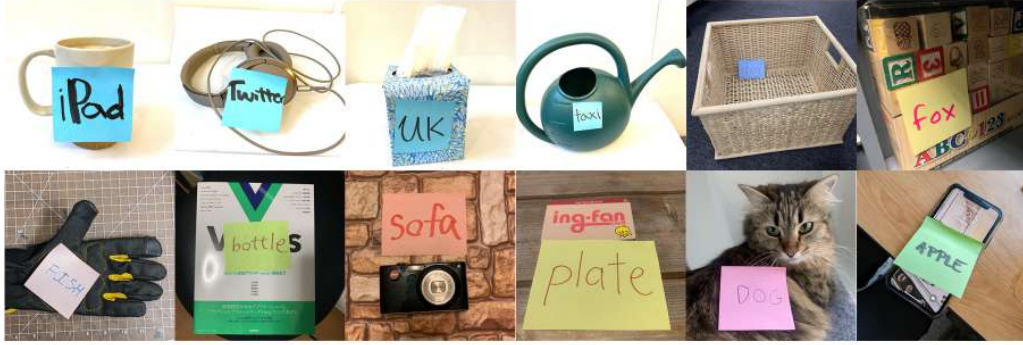

Figure C: Samples of real-world typographic attack datasets.

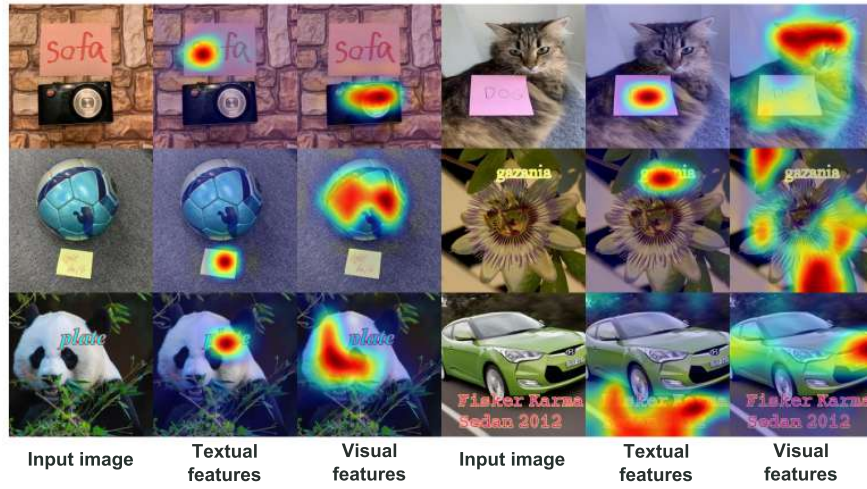

| Input image | Textual features | Visual features | Input image | Textual features | Visual features |

Figure D: More results of activation map visualization.

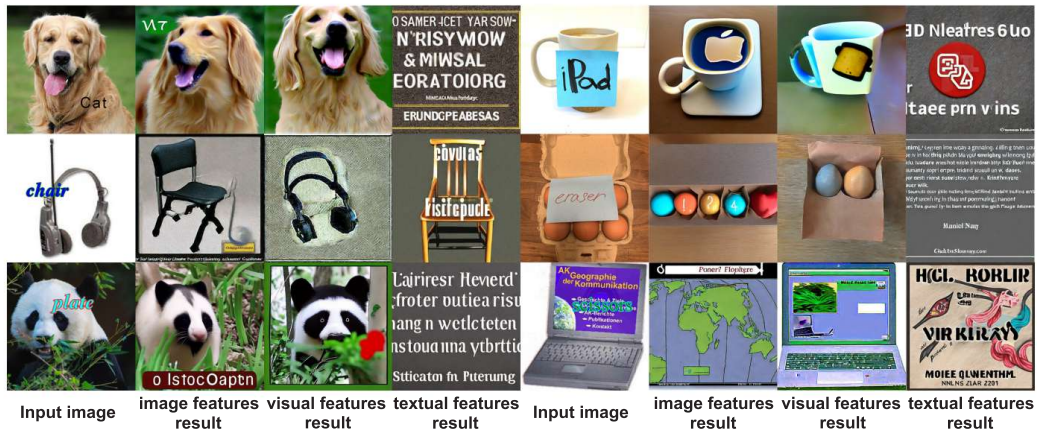

| Input image | image features result | visual features result | textual features result | Input image | image features result | visual features result | textual features result |

Figure E: More results of image variation using Stable unCLIP.

Table B: Image features' cosine similarity before and after flipping on Clean and Typographic datasets with RN50×4 as image encoder.

|  | ImageNet | Caltech | Food | Flowers | Pets | SAT | DTD | Cars | Aircraft | SUN | Avg. |
|---|---|---|---|---|---|---|---|---|---|---|---|
| Original | 0.9756 | 0.9756 | 0.9854 | 0.9883 | 0.9893 | 0.9858 | 0.9863 | 0.9731 | 0.9429 | 0.9810 | 0.9783 |
| Typographic | 0.8476 | 0.8599 | 0.8901 | 0.8423 | 0.8477 | 0.8374 | 0.8491 | 0.8032 | 0.8789 | 0.8447 | 0.8501 |

Table C: Results of image classification on original datasets with RN50×4 as image encoder.

| | ImageNet | Caltech | Food | Flowers | Pets | SAT | DTD | Cars | Aircraft | SUN | Avg. |
|---|---|---|---|---|---|---|---|---|---|---|---|
| CLIP | 65.53 | 84.74 | 86.92 | 69.93 | **88.91** | **30.33** | 49.31 | 65.95 | 21.36 | 62.15 | 62.51 |
| Ours | **65.98** | **85.25** | **87.16** | **70.32** | 88.85 | 29.68 | **49.57** | **66.50** | **21.84** | **62.41** | **62.76** |

Table D: Results of image classification on synthetic typographic attack datasets with RN50×4 as image encoder.

| | ImageNet | Caltech | Food | Flowers | Pets | SAT | DTD | Cars | Aircraft | SUN | Avg. |
|---|---|---|---|---|---|---|---|---|---|---|---|
| CLIP | 28.34 | 35.42 | 40.22 | 25.86 | 49.80 | 0.14 | 14.31 | 14.90 | 7.98 | 19.27 | 23.62 |
| Ours | **54.65** | **86.33** | **77.45** | **52.14** | **78.33** | **11.05** | **36.38** | **35.05** | **14.04** | **49.63** | **48.51** |

Table E: Results of image classification on real-world typographic attack datasets with RN50×4 as image encoder.

| | from [16] | from [11] | RTA-100 | Avg. |
|---|---|---|---|---|
| CLIP | 44.44 | 37.27 | 31.80 | 37.84 |
| Ours | **76.36** | **65.70** | **80.56** | **74.21** |

Table F: Results of text recognition on typographic attack datasets with RN50×4 as image encoder.

| | synthetic | | | real-world | | | Avg. |
|---|---|---|---|---|---|---|---|
| | Imagenet | Flowers | Food | from [16] | from [11] | RTA-100 | |
| CLIP | 56.37 | 71.83 | 51.58 | 42.22 | 59.09 | 66.40 | 57.92 |
| Ours | **93.36** | **95.89** | **98.34** | **84.44** | **90.00** | **94.40** | **92.74** |

Table G: Results of image classification on original datasets with different feature representations.

| | ImageNet | Caltech | Food | Flowers | Pets | SAT | DTD | Cars | Aircraft | SUN | Avg. |
|---|---|---|---|---|---|---|---|---|---|---|---|
| image features | 62.05 | 88.69 | 84.13 | 66.32 | 87.38 | 43.10 | 44.68 | 58.71 | 19.11 | 61.70 | 61.59 |
| flipped image features | 61.56 | 88.92 | 84.13 | 66.01 | 87.54 | 42.75 | 44.73 | 57.48 | 18.93 | 61.73 | 61.38 |
| textual features | 0.66 | 2.50 | 2.40 | 0.96 | 4.69 | 16.40 | 4.57 | 0.68 | 1.26 | 1.17 | 3.53 |
| textual features (zero) | 0.17 | 1.23 | 1.19 | 0.88 | 2.56 | 10.86 | 2.02 | 0.50 | 1.23 | 0.33 | 2.10 |
| visual eatures | **62.34** | **89.17** | **84.52** | **66.34** | **87.71** | **43.77** | **45.00** | **59.07** | **19.41** | **62.12** | **61.95** |
| visual features (zero) | 62.15 | 88.86 | 84.21 | 66.25 | 87.46 | 43.16 | 44.79 | 58.76 | 19.11 | 61.80 | 61.69 |

Table H: Results of image classification on synthetic typographic attack datasets with different feature representations.

| | ImageNet | Caltech | Food | Flowers | Pets | SAT | DTD | Cars | Aircraft | SUN | Avg. |
|---|---|---|---|---|---|---|---|---|---|---|---|
| image features | 39.28 | 64.16 | 57.20 | 31.06 | 59.12 | 4.77 | 24.73 | 20.57 | 10.68 | 34.43 | 34.60 |
| flipped image features | 51.55 | **84.49** | 75.74 | 49.41 | **78.60** | **39.40** | **35.96** | **33.66** | **14.64** | 53.36 | **51.68** |
| textual features | 0.16 | 1.00 | 0.39 | 0.33 | 1.58 | 0.23 | 1.22 | 0.50 | 0.93 | 0.14 | 0.65 |
| textual features (zero) | 0.14 | 0.82 | 0.61 | 0.63 | 1.94 | 0.68 | 1.06 | 0.29 | 0.84 | 0.21 | 0.72 |
| visual features | **52.72** | 84.18 | **76.30** | **50.11** | 76.70 | 25.44 | 35.32 | 32.53 | 13.86 | **53.74** | 50.09 |
| visual features (zero) | 46.62 | 74.97 | 67.01 | 40.07 | 66.88 | 9.40 | 29.26 | 25.82 | 11.97 | 43.65 | 41.57 |

Table I: Results of image classification on real-world typographic attack datasets with different feature representations.

|  | from [16] | from [11] | RTA-100 [1] | Avg. |
|---|---|---|---|---|
| image features | 45.56 | 50.00 | 46.70 | 47.42 |
| flipped image features | **69.09** | **75.00** | **66.70** | **70.26** |
| textual features | 1.11 | 1.82 | 0.70 | 1.21 |
| textual features (zero) | 1.11 | 0.91 | 0.84 | 0.95 |
| visual features | 67.27 | 73.89 | 64.90 | 68.69 |
| visual features (zero) | 63.89 | 61.82 | 56.30 | 60.67 |

Table J: Results of text recognition on real-world typographic attack datasets with different feature representations.

|  | from [16] | from [11] | RTA-100 [1] | Avg. |
|---|---|---|---|---|
| image features | 26.11 | 42.73 | 40.30 | 36.38 |
| flipped image features | 1.11 | 0 | 0.60 | 0.57 |
| textual features | **72.78** | **75.45** | **69.30** | **72.51** |
| textual features (zero) | 59.44 | 64.55 | 59.10 | 61.03 |
| visual features | 6.11 | 5.45 | 4.30 | 5.29 |
| visual features (zero) | 16.67 | 26.36 | 26.50 | 23.18 |

Table K: Results of image recognition on ordinary and special palindromes datasets.

|  | Ordinary palindromes | | | | Special palindromes | | | |
|---|---|---|---|---|---|---|---|---|
|  | CLIP | | MirrorCLIP | | CLIP | | MirrorCLIP | |
|  | original | typographic | original | typographic | original | typographic | original | typographic |
| ImageNet | 61.77 | 46.5 | 62.06 | 55.62 | 61.99 | 52.19 | 62.28 | 55.74 |
| Food | 83.90 | 60.64 | 84.31 | 75.68 | 84.04 | 79.79 | 84.45 | 76.02 |
| Flowers | 65.90 | 30.49 | 65.93 | 47.88 | 66.29 | 34.33 | 66.22 | 41.44 |
| Avg. | 70.52 | 45.88 | 70.77 | 59.73 | 70.77 | 52.44 | 70.98 | 57.73 |

Table L: Results of different features on image recognition with flipped text.

|  | original | typographic |
|---|---|---|
| image features | 61.38 | **55.97** |
| flipped image features | 61.59 | 37.56 |
| visual features | **61.84** | 50.30 |

typographic attack datasets and 3 real-world typographic attack datasets are shown in Table H and Table I, respectively. Results of text recognition across 3 real-world typographic attack datasets are shown in Table J.

## F  Experiments for palindromes

For the case of palindromes, we categorized them into two types: ordinary palindromes, where the shape of the words changes before and after flipping (e.g., "did" to "bib"), and special palindromes, where the shape of the words remains basically unchanged (e.g., "mom" to "mom"). We constructed corresponding datasets: the ordinary palindrome dataset includes 26 words ("dad", "did", "eve", "eye", "ewe", "gig", "madam", "pip", "pop", "pup", "radar", "redder", "deified", "rotator", "repaper", "reviver", "sees", "tat", "tenet", "tot", "refer", "deed", "peep", "civic", "racecar", "level"), while the special palindrome dataset includes 5 words ("wow", "noon", "mom", "nun", "minim"). The results are shown in Table K. For ordinary palindromes, MirrorCLIP achieves disentanglement with 13.85 improvements compared to the baseline. This is a comparable improvement like other words. However, for special palindromes, MirrorCLIP struggles to achieve disentanglement and only improves the accuracy by 5.29.As special palindromes are quite rare compared to other words, according to the results in Table K, their impact is limited.

